# Selecting weighting factors in logarithmic opinion pools

**Tom Heskes**
Foundation for Neural Networks, University of Nijmegen
Geert Grooteplein 21, 6525 EZ Nijmegen, The Netherlands
tom@mbfys.kun.nl

## Abstract

A simple linear averaging of the outputs of several networks as e.g. in bagging [3], seems to follow naturally from a bias/variance decomposition of the sum-squared error. The sum-squared error of the average model is a quadratic function of the weighting factors assigned to the networks in the ensemble [7], suggesting a quadratic programming algorithm for finding the "optimal" weighting factors.

If we interpret the output of a network as a probability statement, the sum-squared error corresponds to minus the loglikelihood or the Kullback-Leibler divergence, and linear averaging of the outputs to logarithmic averaging of the probability statements: the logarithmic opinion pool.

The crux of this paper is that this whole story about model averaging, bias/variance decompositions, and quadratic programming to find the optimal weighting factors, is not specific for the sum-squared error, but applies to the combination of probability statements of any kind in a logarithmic opinion pool, as long as the Kullback-Leibler divergence plays the role of the error measure. As examples we treat model averaging for classification models under a cross-entropy error measure and models for estimating variances.

## 1 INTRODUCTION

In many simulation studies it has been shown that combining the outputs of several trained neural networks yields better results than relying on a single model. For regression problems, the most obvious combination seems to be a simple linear

averaging of the network outputs. From a bias/variance decomposition of the sum-squared error it follows that the error of the so obtained average model is always smaller or equal than the average error of the individual models. In [7] simple linear averaging is generalized to weighted linear averaging, with different weighting factors for the different networks in the ensemble. A slightly more involved bias/variance decomposition suggests a rather straightforward procedure for finding "optimal" weighting factors.

Minimizing the sum-squared error is equivalent to maximizing the loglikelihood of the training data under the assumption that a network output can be interpreted as an estimate of the mean of a Gaussian distribution with fixed variance. In these probabilistic terms, a linear averaging of network outputs corresponds to a logarithmic rather than linear averaging of probability statements.

In this paper, we generalize the regression case to the combination of probability statements of any kind. Using the Kullback-Leibler divergence as the error measure, we naturally arrive at the so-called logarithmic opinion pool. A bias/variance decomposition similar to the one for sum-squared error then leads to an objective method for selecting weighting factors.

Selecting weighting factors in any combination of probability statements is known to be a difficult problem for which several suggestions have been made. These suggestions range from rather involved supra-Bayesian methods to simple heuristics (see e.g. [1, 6] and references therein). The method that follows from our analysis is probably somewhere in the middle: easier to compute than the supra-Bayesian methods and more elegant than simple heuristics.

To stress the generality of our results, the presentation in the next section will be rather formal. Some examples will be given in Section 3. Section 4 discusses how the theory can be transformed into a practical procedure.

## 2   LOGARITHMIC OPINION POOLS

Let us consider the general problem of building a probability model of a variable $y$ given a particular input $x$. The "output" $y$ may be continuous, as for example in regression analysis, or discrete, as for example in classification. In the latter case integrals over $y$ should be replaced by summations over all possible values of $y$. Both $x$ and $y$ may be vectors of several elements; the one-dimensional notation is chosen for convenience. We suppose that there is a "true" conditional probability model $q(y|x)$ and have a whole ensemble (also called pool or committee) of experts, each supplying a probability model $p_\alpha(y|x)$. $\rho(x)$ is the unconditional probability distribution of inputs. An unsupervised scenario, as for example treated in [8], is obtained if we simply neglect the inputs $x$ or consider them constant.

We define the distance between the true probability $q(y|x)$ and an estimate $p(y|x)$ to be the Kullback-Leibler divergence

$$K(q,p) \equiv - \int dx\, \rho(x) \int dy\, q(y|x) \log \left[ \frac{p(y|x)}{q(y|x)} \right] .$$

If the densities $\rho(x)$ and $q(y|x)$ correspond to a data set containing a finite number $P$ of combinations $\{x^\mu, y^\mu\}$, minus the Kullback divergence is, up to an irrelevant

constant, equivalent to the loglikelihood defined as

$$L(p, \{\vec{x}, \vec{y}\}) \equiv \frac{1}{P} \sum_{\mu} \log p(y^{\mu} | x^{\mu}) \, .$$

The more formal use of the Kullback-Leibler divergence instead of the loglikelihood is convenient in the derivations that follow.

Weighting factors $w_{\alpha}$ are introduced to indicate the reliability of each of the experts $\alpha$. In the following we will work with the constraints $\sum_{\alpha} w_{\alpha} = 1$, which is used in some of the proofs, and $w_{\alpha} \geq 0$ for all experts $\alpha$, which is not strictly necessary, but makes it easier to interpret the weighting factors and helps to prevent overfitting when weighting factors are optimized (see details below).

We define the average model $\bar{p}(y|x)$ to be the one that is closest to the given set of models:

$$\bar{p}(y|x) \equiv \operatorname*{argmin}_{p(y|x)} \sum_{\alpha} w_{\alpha} K(p, p_{\alpha}) \, .$$

Introducing a Lagrange multiplier for the constraint $\int dx p(y|x) = 1$, we immediately find the solution

$$\bar{p}(y|x) = \frac{1}{Z(x)} \prod_{\alpha} [p_{\alpha}(y|x)]^{w_{\alpha}} \, , \tag{1}$$

with normalization constant

$$Z(x) = \int dy \prod_{\alpha} [p_{\alpha}(y|x)]^{w_{\alpha}} \, . \tag{2}$$

This is the logarithmic opinion pool, to be contrasted with the linear opinion pool, which is a linear average of the probabilities. In fact, logarithmic opinion pools have been proposed to overcome some of the weaknesses of the linear opinion pool. For example, the logarithmic opinion pool is "externally Bayesian", i.e., can be derived from joint probabilities using Bayes' rule [2]. A drawback of the logarithmic opinion pool is that if any of the experts assigns probability zero to a particular outcome, the complete pool assigns probability zero, no matter what the other experts claim. This property of the logarithmic opinion pool, however, is only a drawback if the individual density functions are not carefully estimated. The main problem for both linear and logarithmic opinion pools is how to choose the weighting factors $w_{\alpha}$.

The Kullback-Leibler divergence of the opinion pool $\bar{p}(y|x)$ can be decomposed into a term containing the Kullback-Leibler divergences of individual models and an "ambiguity" term:

$$K(q, \bar{p}) = \sum_{\alpha} w_{\alpha} K(q, p_{\alpha}) - \sum_{\alpha} w_{\alpha} K(\bar{p}, p_{\alpha}) \equiv E - A \, . \tag{3}$$

**Proof:** The first term in (3) follows immediately from the numerator in (1), the second term is minus the logarithm of the normalization constant $Z(x)$ in (2) which can, using (1), be rewritten as

$$A = \int dx \, \rho(x) \log[Z(x)] = \int dx \rho(x) \log \left[ \frac{\prod_{\alpha} [p_{\alpha}(y'|x)]^{w_{\alpha}}}{\bar{p}(y'|x)} \right] \, ,$$

for any choice of $y'$ for which $\bar{p}(y'|x)$ is nonzero. Integration over $y'$ with probability measure $\bar{p}(y'|x)$ then yields (3).

Since the ambiguity $A$ is always larger than or equal to zero, we conclude that the Kullback-Leibler divergence of the logarithmic opinion pool is never larger than the average Kullback-Leibler divergences of individual experts. The larger the ambiguity, the larger the benefit of combining the experts' probability assessments. Note that by using Jensen's inequality, it is also possible to show that the Kullback-Leibler divergence of the linear opinion pool is smaller or equal to the average Kullback-Leibler divergences of individual experts. The expression for the ambiguity, defined as the difference between these two, is much more involved and more difficult to interpret (see e.g. [10]).

The ambiguity of the logarithmic opinion pool depends on the weighting factors $w_\alpha$, not only directly as expressed in (3), but also through $\bar{p}(y|x)$. We can make this dependency somewhat more explicit by writing

$$ A = \frac{1}{2} \sum_{\alpha\beta} w_\alpha w_\beta K(p_\alpha, p_\beta) + \frac{1}{2} \sum_{\alpha} w_\alpha \left[ K(\bar{p}, p_\alpha) - K(p_\alpha, \bar{p}) \right] . \tag{4} $$

**Proof:** Equation (3) is valid for any choice of $q(y|x)$. Substitute $q(y|x) = p_\beta(y|x)$, multiply left- and righthand side by $w_\beta$, and sum over $\beta$. Simple manipulation of terms than yields the result.

Alas, the Kullback-Leibler divergence is not necessarily symmetric, i.e., in general $K(p_1, p_2) \neq K(p_2, p_1)$. However, the difference $K(p_1, p_2) - K(p_2, p_1)$ is an order of magnitude smaller than the divergence $K(p_1, p_2)$ itself. More formally, writing $p_1(y|x) = [1+\epsilon(y|x)]p_2(y|x)$ with $\epsilon(y|x)$ small, we can easily show that $K(p_1, p_2)$ is of order (some integral over) $\epsilon^2(y|x)$ whereas $K(p_1, p_2) - K(p_2, p_1)$ is of order $\epsilon^3(y|x)$. Therefore, if we have reason to assume that the different models are reasonably close together, we can, in a first approximation, and will, to make things tractable, neglect the second term in (4) to arrive at

$$ K(q, \bar{p}) \approx \sum_{\alpha} w_\alpha K(q, p_\alpha) - \frac{1}{4} \sum_{\alpha,\beta} w_\alpha w_\beta \left[ K(p_\alpha, p_\beta) + K(p_\beta, p_\alpha) \right] . \tag{5} $$

The righthand side of this expression is quadratic in the weighting factors $w_\alpha$, a property which will be very convenient later on.

## 3  EXAMPLES

**Regression.** The usual assumption in regression analysis is that the output functionally depends on the input $x$, but is blurred by Gaussian noise with standard deviation $\sigma$. In other words, the probability model of an expert $\alpha$ can be written

$$ p_\alpha(y|x) = \sqrt{\frac{1}{2\pi\sigma^2}} \exp\left[ \frac{-(y - f_\alpha(x))^2}{2\sigma^2} \right] . \tag{6} $$

The function $f_\alpha(x)$ corresponds to the network's estimate of the "true" regression given input $x$. The logarithmic opinion pool (1) also leads to a normal distribution with the same standard deviation $\sigma$ and with regression estimate

$$ \bar{f}(x) = \sum_{\alpha} w_\alpha f_\alpha(x) . $$

In this case the Kullback-Leibler divergence

$$K(p_\alpha, p_\beta) = \frac{1}{2\sigma^2} \int dx\, \rho(x)\, [f_\alpha(x) - f_\beta(x)]^2$$

is symmetric, which makes (5) exact instead of an approximation. In [7], this has all been derived starting from a sum-squared error measure.

**Variance estimation.** There has been some recent interest in using neural networks not only to estimate the mean of the target distribution, but also its variance (see e.g. [9] and references therein). In fact, one can use the probability density (6) with input-dependent $\sigma(x)$. We will consider the simpler situation in which an input-dependent model is fitted to residuals $y$, *after* a regression model has been fitted to estimate the mean (see also [5]). The probability model of expert $\alpha$ can be written

$$p_\alpha(y|x) = \sqrt{\frac{z_\alpha(x)}{2\pi}} \exp\left[-\frac{z_\alpha(x)y^2}{2}\right],$$

where $1/z_\alpha(x)$ is the experts' estimate of the residual variance given input $x$. The logarithmic opinion pool is of the same form with $z_\alpha(x)$ replaced by

$$\bar{z}(x) = \sum_\alpha w_\alpha z_\alpha(x).$$

Here the Kullback-Leibler divergence

$$K(\bar{p}, p_\alpha) = \frac{1}{2} \int dx\, \rho(x) \left[\frac{\bar{z}(x)}{z_\alpha(x)} - \log\frac{\bar{z}(x)}{z_\alpha(x)} - 1\right]$$

is asymmetric. We can use (3) to write the Kullback-Leibler divergence of the opinion pool explicitly in terms of the weighting factors $w_\alpha$. The approximation (5), with

$$K(p_\alpha, p_\beta) + K(p_\beta, p_\alpha) = \frac{1}{2} \int dx\, \rho(x)\, \frac{[z_\alpha(x) - z_\beta(x)]^2}{z_\alpha(x) z_\beta(x)},$$

is much more appealing and easier to handle.

**Classification.** In a two-class classification problem, we can treat $y$ as a discrete variable having two possible realizations, e.g., $y \in \{-1, 1\}$. A convenient representation for a properly normalized probability distribution is

$$p_\alpha(y|x) = \frac{1}{1 + \exp[-2h_\alpha(x)y]}.$$

In this logistic representation, the logarithmic opinion pool has the same form with

$$\bar{h}(x) = \sum_\alpha w_\alpha h_\alpha(x).$$

The Kullback-Leibler divergence is asymmetric, but yields the simpler form

$$K(p_\alpha, p_\beta) + K(p_\beta, p_\alpha) = \int dx\, \rho(x)\, [\tanh(h_\alpha(x)) - \tanh(h_\beta(x))][h_\alpha(x) - h_\beta(x)],$$

to be used in the approximation (5). For a finite set of patterns, minus the loglikelihood yields the well-known cross-entropy error.

The probability models in these three examples are part of the exponential family. The mean $f_\alpha$, inverse variance $z_\alpha$, and logit $h_\alpha$ are the canonical parameters. It is straightforward to show that, with constant dispersion across the various experts, the canonical parameter of the logarithmic opinion pool is always a weighted average of the canonical parameters of the individual experts. Slightly more complicated expressions arise when the experts are allowed to have different estimates for the dispersion or for probability models that do not belong to the exponential family.

## 4  SELECTING WEIGHTING FACTORS

The decomposition (3) and approximation (5) suggest an objective method for selecting weighting factors in logarithmic opinion pools. We will sketch this method for an ensemble of models belonging to the same class, say feedforward neural networks with a fixed number of hidden units, where each model is optimized on a different bootstrap replicate of the available data set.

Suppose that we have available a data set consisting of $P$ combinations $\{x^\mu, y^\mu\}$. As suggested in [3], we construct different models by training them on different bootstrap replicates of the available data set. Optimizing nonlinear models is often an unstable process: small differences in initial parameter settings or two almost equivalent bootstrap replicates can result in completely different models. Neural networks, for example, are notorious for local minima and plateaus in weight space where models might get stuck. Therefore, the incorporation of weighting factors, even when models are constructed using the same procedure, can yield a better generalizing opinion pool. In [4] good results have been reported on several regression problems. Balancing clearly outperformed bagging, which corresponds to $w_\alpha = 1/n$ with $n$ the number of experts, and bumping, which proposes to keep a single expert.

Each example in the available data set can be viewed as a realization of an unknown probability density characterized by $\rho(x)$ and $q(y|x)$. We would like to choose the weighting factors $w_\alpha$ such as to minimize the Kullback-Leibler divergence $K(q, \bar{p})$ of the opinion pool. If we accept the approximation (5), we can compute the optimal weighting factors once we know the individual Kullbacks $K(q, p_\alpha)$ and the Kullbacks between different models $K(p_\alpha, p_\beta)$. Of course, both $q(y|x)$ and $\rho(x)$ are unknown, and thus we have to settle for estimates.

In an estimate for $K(p_\alpha, p_\beta)$ we can simply replace the average over $\rho(x)$ by an average over all inputs $x^\mu$ observed in the data set:

$$K(p_\alpha, p_\beta) + K(p_\beta, p_\alpha) \approx \frac{1}{P} \sum_\mu \int dy\, [p_\alpha(y|x^\mu) - p_\beta(y|x^\mu)] \log \left[ \frac{p_\alpha(y|x^\mu)}{p_\beta(y|x^\mu)} \right] .$$

A similar straightforward replacement for $q(y|x)$ in an estimate for $K(q, p_\alpha)$ is biased, since each expert has, at least to some extent, been overfitted on the data set. In [4] we suggest how to remove this bias for regression models minimizing sum-squared errors. Similar compensations can be found for other probability models.

Having estimates for both the individual Kullback-Leibler divergences $K(q, p_\alpha)$ and the cross terms $K(p_\alpha, p_\beta)$, we can optimize for the weighting factors $w_\alpha$. Under the constraints $\sum_\alpha w_\alpha = 1$ and $w_\alpha \geq 0$ the approximation (5) leads to a quadratic programming problem. Without this approximation, optimizing the weighting factors becomes a nasty exercise in nonlinear programming.

The solution of the quadratic programming problem usually ends up at the edge of the unit cube with many weighting factors equal to zero. On the one hand, this is a beneficial property, since it implies that we only have to keep a relatively small number of models for later processing. On the other hand, the obtained weighting factors may depend too strongly on our estimates of the individual Kullbacks $K(q, p_\alpha)$. The following version prohibits this type of overfitting. Using simple statistics, we obtain a rough indication for the accuracy of our estimates $K(q, p_\alpha)$. This we use to generate several, say on the order of 20, different samples with estimates $\{K(q, p_1), \ldots, K(q, p_n)\}$. For each of these samples we solve the corresponding quadratic programming problem and obtain a set of weighting factors. The final weighting factors are obtained by averaging. In the end, there are less experts with zero weighting factors, at the advantage of a more robust procedure.

## Acknowledgements

I would like to thank David Tax, Bert Kappen, Piërre van de Laar, Wim Wiegerinck, and the anonymous referees for helpful suggestions. This research was supported by the Technology Foundation STW, applied science division of NWO and the technology programme of the Ministry of Economic Affairs.

# References

[1] J. Benediktsson and P. Swain. Consensus theoretic classification methods. *IEEE Transactions on Systems, Man, and Cybernetics*, 22:688–704, 1992.

[2] R. Bordley. A multiplicative formula for aggregating probability assessments. *Management Science*, 28:1137–1148, 1982.

[3] L. Breiman. Bagging predictors. *Machine Learning*, 24:123–140, 1996.

[4] T. Heskes. Balancing between bagging and bumping. In M. Mozer, M. Jordan, and T. Petsche, editors, *Advances in Neural Information Processing Systems 9*, pages 466–472, Cambridge, 1997. MIT Press.

[5] T. Heskes. Practical confidence and prediction intervals. In M. Mozer, M. Jordan, and T. Petsche, editors, *Advances in Neural Information Processing Systems 9*, pages 176–182, Cambridge, 1997. MIT Press.

[6] R. Jacobs. Methods for combining experts' probability assessments. *Neural Computation*, 7:867–888, 1995.

[7] A. Krogh and J. Vedelsby. Neural network ensembles, cross validation, and active learning. In G. Tesauro, D. Touretzky, and T. Leen, editors, *Advances in Neural Information Processing Systems 7*, pages 231–238, Cambridge, 1995. MIT Press.

[8] P. Smyth and D. Wolpert. Stacked density estimation. *These proceedings*, 1998.

[9] P. Williams. Using neural networks to model conditional multivariate densities. *Neural Computation*, 8:843–854, 1996.

[10] D. Wolpert. On bias plus variance. *Neural Computation*, 9:1211–1243, 1997.